# A Neural Expert System with Automated Extraction of Fuzzy If-Then Rules and Its Application to Medical Diagnosis

Yoichi Hayashi*
Department of Computer and Information Sciences
Ibaraki University
Hitachi-shi, Ibaraki 316, Japan

## ABSTRACT

This paper proposes a *fuzzy neural expert system* (FNES) with the following two functions: (1) Generalization of the information derived from the training data and embodiment of knowledge in the form of the fuzzy neural network; (2) Extraction of fuzzy If-Then rules with *linguistic relative importance* of each proposition in an antecedent (If-part) from a trained neural network. This paper also gives a method to extract automatically fuzzy If-Then rules from the trained neural network. To prove the effectiveness and validity of the proposed fuzzy neural expert system, a fuzzy neural expert system for medical diagnosis has been developed.

## 1 INTRODUCTION

Expert systems that have neural networks for their knowledge bases are sometimes called *neural expert system* (Gallant & Hayashi, 1990; Hayashi et al., 1990; Yoshida et al., 1990) or *connectionist expert system* (Gallant, 1988; Yoshida et al., 1989). This paper extends work reported in (Hayashi & Nakai, 1990; Hayashi et al., 1990) and shows a new method to give confidence measurements for all inferences and explanations to neural expert systems. In contrast with conventional expert systems, we propose a *fuzzy neural expert system* (FNES) with automated extraction of fuzzy If-Then rules. This paper also gives a method to extract automatically fuzzy If-Then rules with *linguistic relative importance* of each proposition in an antecedent (If-part) from a trained neural network. To prove the effectiveness and validity of the proposed neural expert system, a fuzzy neural expert system for diagnosing hepatobiliary disorders has been developed by using a real medical database. This paper compares the diagnostic capability provided by the neural network approach and that provided by the statistical approach. Furthermore, we evaluate the performance of extracted fuzzy If-Then rules from a neural network knowledge base.

## 2  FUZZY NEURAL EXPERT SYSTEM WITH AUTOMATED EXTRACTION OF FUZZY IF-THEN RULES

### 2.1  Distributed Neural Network

Figure 1 illustrates a schematic diagram of a fuzzy neural expert system with automated extraction of fuzzy If-Then rules. For backpropagation, the configuration consisting of $p$ input cells, $q$ intermediate cells ("hidden units") and $r$ output cells has been the most widely used. Connections run from every input cell to every intermediate cell, and from every intermediate cell to every output cell. In this paper, we employ a valiant of conventional perceptron network, which is called *distributed (neural) network* (Gallant, 1990). In the network, there are the same cells and connections as with the backpropagation, and in addition there are direct connections from input to output cells. See Figure 2. Each connection has an integer weight $w_{ij}$ that roughly corresponds to the influence of cell $u_j$ on cell $u_i$. Although the weights of connections from the input layer to the intermediate layer are generated by using a random number generator (in this paper, integers between -10 and +10 were used) and fixed for learning process. Cell activations are discrete, each taking on values +1, 0, or -1.

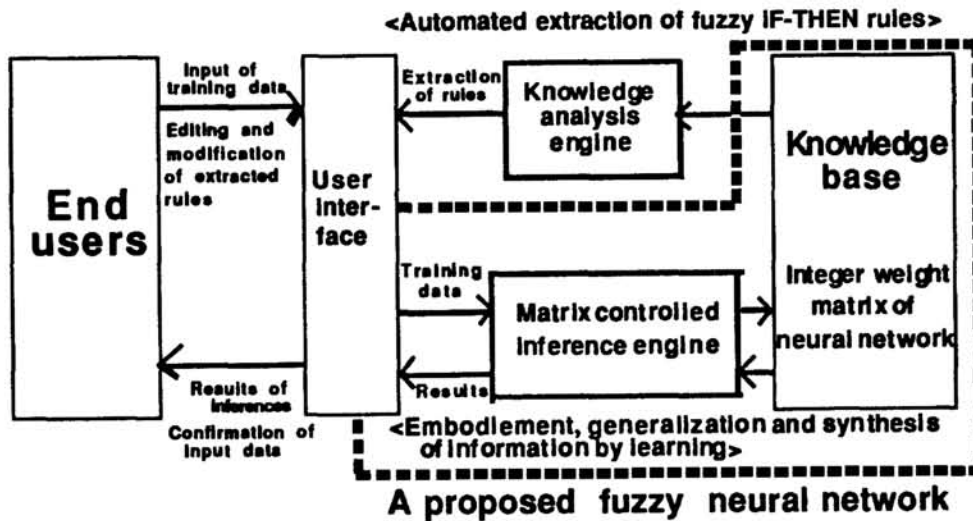

**Figure 1: A Schematic Diagram of A Fuzzy Neural System with Automated Extraction of Fuzzy IF-THEN Rules**

Activations of the input cells $I_i$ ($i = 1,2,...,p$), the intermediate cell $H_j$ ($j = 1,2,...,q$) and the output cell $O_k$ ($k = 1,2,...,r$) can be calculated using equations (1) - (4). The value of the cell $I_0$ is always +1, and it is connected to every other cell except for input cells.

$$SH_j = \sum_{i=0}^{p} w_{ji} I_i \qquad (1)$$

$$SO_k = \sum_{i=0}^{p} u_{ki} I_i + \sum_{j=1}^{q} v_{kj} H_j \qquad (3)$$

$$H_j = \begin{cases} +1 \ or \ True & (SH_j > 0) \\ 0 \ or \ Unknown & (SH_j = 0) \\ -1 \ or \ False & (SH_j < 0) \end{cases} \quad (2)$$

$$O_k = \begin{cases} +1 \ or \ True & (SO_k > 0) \\ 0 \ or \ Unknown & (SO_k = 0) \\ -1 \ or \ False & (SO_k < 0) \end{cases} \quad (4)$$

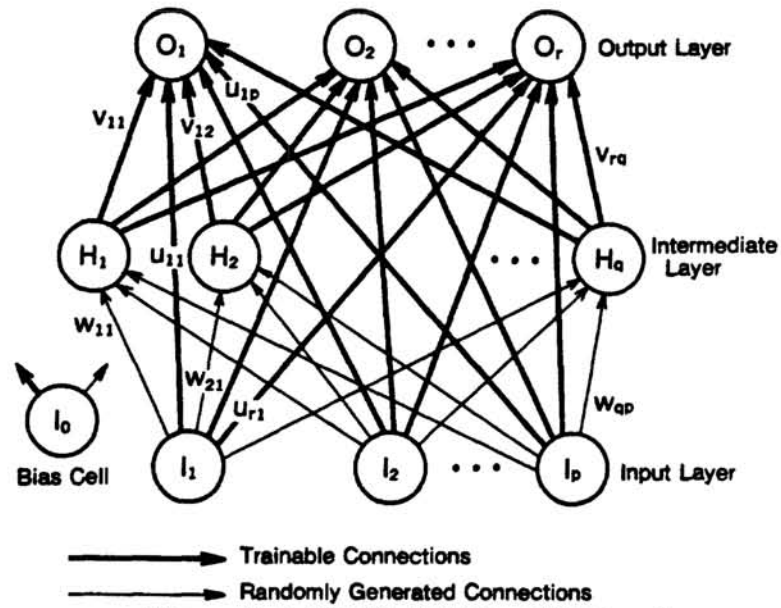

Figure 2: A Distributed Neural Network

## 2.2  Fuzzy Neural Network

To handle various fuzziness in the input layer of the distributed neural network, it is necessary to interpret  subjective input data which has non-Boolean quantitative and/or qualitative meaning. In general, fuzzy sets defined by monotone membership functions can be "defuzzified" into a family of crisp sets by using the level set representation (Negoita, 1985) or "thermometer code" of B. Widrow. Therefore, the fuzziness can be incorporated into the training data by using only Boolean inputs.  Once the training data is set up in this manner, it can be processed by the Pocket Algorithm (Gallant, 1990). In this paper,  we will propose a *fuzzy neural network* to handle  fuzzy  data and  crisp  data

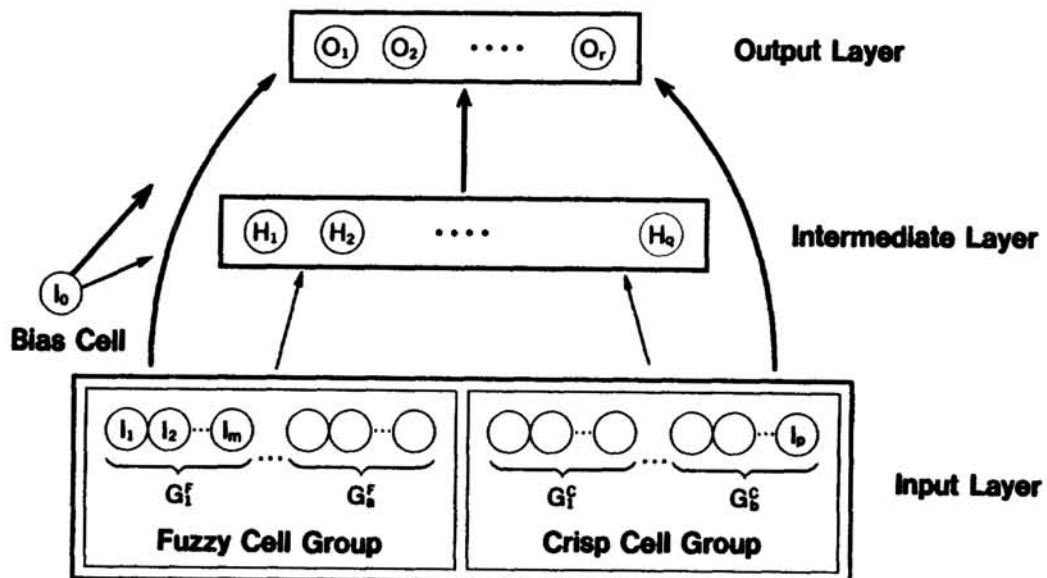

Figure 3: A Neural Network with Fuzzy Cell Groups and Crisp Cell Groups

given in the input layer.  Figure 3 shows the structure of proposed fuzzy neural network whose input layer consists of fuzzy cell groups and crisp (non-fuzzy) cell groups. Here,

truthfulness of fuzzy information and crisp information such as binary encoded data is represented by fuzzy cell groups and crisp cell groups, respectively. A fuzzy cell group consists of $m$ input cells which have the level set  representation using binary $m$-dimensional vector, each taking on values in $\{+1, -1\}$; whereas a crisp cell group also consists of  $m$ input cells which take on two values in $\{(+1,+1,...,+1), (-1,-1,...,-1)\}$.

## 3  AUTOMATED EXTRACTION OF FUZZY IF-THEN RULES FROM TRAINED NEURAL NETWORKS

This paper also extends previous work described in (Hayashi & Nakai, 1990) and proposes a method to extract automatically fuzzy If-Then rules with *linguistic relative importance* of each proposition in an antecedent (Hayashi & Nakai, 1989) from a trained fuzzy neural network.  The method is implemented in the knowledge analysis engine in Figure 1.  The linguistic relative importance such as *Very Important* and *Moderately Important*, which is defined by a fuzzy set, represents the degree of effect of each proposition on consequence.  By providing linguistic relative importance for each proposition, each fuzzy If-Then rule has more flexible expression than that of ordinary If-Then rules.  Furthermore, truthfulness of each fuzzy If-Then rule is given in the form of linguistic truth value such as *Very True* and *Possibly True*, which is defined by a fuzzy set.  Enhancement of the presentation capability and flexibility by using fuzzy If-Then rules with linguistic relative importance facilitates the automated extraction of fuzzy If-Then rules from a trained neural network.

### 3.1  Automated If-Then Rule Extraction Algorithm
We have proposed some methods to extract fuzzy If-Then rules with linguistic relative importance from a trained (fuzzy) neural network.  In this section, we extend work reported in (Hayashi & Nakai, 1990; Hayashi et al., 1990) and give an algorithm to extract fuzzy If-Then rules  from a trained fuzzy neural network in the following.  Note that an exact algorithm of Step 2 and Step 3 can be derived from algorithms shown in (Hayashi & Nakai, 1990) in the same manner.  Here, we will give a brief discussion on them due to space limitation.  We shall concentrate on Step1.
**Step 1.  Extraction of framework of fuzzy If-then rules:** We  select propositions in an antecedent (If-part) of a rule, that is, extract framework of fuzzy If-Then rules.  We will give a precise algorithm for this step in section 3.2.
**Step 2. Assignment of linguistic truth value to each  extracted rule:** A linguistic truth value such as *Very Very True* (V.V.T.) and *Possibly True* (P.T.) is given to each fuzzy If-Then rule selected in Step 1.  Linguistic truth value assigned to each rule indicates the degree of certainty to draw the conclusion.  The linguistic truth value is determined by the relative amount of weighted sum of output cells.
**Step 3.  Assignment of linguistic relative importance  to each proposition:** Linguistic relative importance is assigned to each proposition of antecedent in fuzzy If-Then rules. Linguistic relative importance such as *Very Important* (V.I.) and *Moderately Important* (M.I.) represents the degree of effect of each proposition on consequence.

### 3.2  Algorithm to extract framework of fuzzy If-Then rules
Extraction of dispensable propositions on cell groups in an antecedent (If-part) is required for the extraction of framework of fuzzy If-Then rules.  For simplicity,  it is

supposed in this section that each cell consists of three input cells. Therefore, a fuzzy cell group takes on three values in {(+1,-1,-1), (+1,+1,-1), (+1,+1,+1)}; whereas a crisp cell group takes on two values in {(+1,+1,+1), (-1,-1,-1)}. In distributed neural network, we can determine activations (values) of cells using partial input information. For example, activations of intermediate cell $H_j$ are determined as

$$H_j = \begin{cases} +1 \quad or \quad True & ( \ |SH_j| > USH_j \ \ and \ \ SH_j > 0 \ ) \\ 0 \quad or \quad Unknown & ( \ |SH_j| \le USH_j ) \\ -1 \quad or \quad False & ( \ |SH_j| > USH_j \ \ and \ \ SH_j < 0 \ ) \end{cases} \qquad (5)$$

where

$$USH_j = \sum_{j \,:\, I_i \text{ is } Unknown} |w_{ji}| . \qquad (6)$$

In the same manner, activations of output cell $O_k$ are determined as

$$O_k = \begin{cases} +1 \quad or \quad True & ( \ |SO_k| > USO_k \ \ and \ \ SO_k > 0 \ ) \\ 0 \quad or \quad Unknown & ( \ |SO_k| \le USO_k ) \\ -1 \quad or \quad False & ( \ |SO_k| > USO_k \ \ and \ \ SH_k < 0 \ ) \end{cases} \qquad (7)$$

where

$$USO_k = \sum_{i \,:\, I_i \text{ is } Unknown} |u_{ki}| + \sum_{j \,:\, H_j \text{ is } Unknown} |v_{kj}| . \qquad (8)$$

Our problem is to determine the value of input cell groups so that each output cell $O_k$ takes on values +1 or -1. Propositions (Input items) corresponding to determined input cell groups will be entrapped in an antecedent (If-part) of each rule. We will give an extraction algorithm for framework of fuzzy If-Then rules as follows:

**Step I:** Select one output cell $O_k$.

**Step II:** Select one cell group. If the selected cell group is a fuzzy cell group, set the values of the cell group in (+1,-1,-1), (+1,+1,-1) or (+1,+1,+1); whereas if the selected cell group is a crisp cell group, set the values of the cell group in (+1,+1,+1) or (-1,-1,-1). Furthermore, set the value of cell groups which were not selected to (0, 0, 0).

**Step III (Forward search):** Determine all the value of intermediate cells $H_j$ by using the values of cell groups given in Step II and equation (5). Furthermore, determine the value of output cell $O_k$ using (7). If the value of $O_k$ is +1 or -1, go to Step V. Otherwise (the value of $O_k$ is 0), go to Step IV. Although all the cell groups are entrapped in an antecedent (If-part), if the value of $O_k$ is 0, there is no framework of fuzzy If-Then rules for the output cell $O_k$ and go to Step VI.

**Step IV (Backward search):** Let $v^*$ be the maximum value of $|v_{kj}|$ which is an absolute value of the weight of the connections between the output cell $O_k$ and the intermediate cell $H_j$ whose activation value is 0. Furthermore, let $u^*$ be the maximum value of $|u_{ki}|$ which is an absolute value of the weight of the connections between the output cell $O_k$ and the input cell $I_i$ whose value is 0. If $u^* \ge v^*$ or values of all the intermediate cells are determined, go to Step IV-1. Otherwise, go to Step IV-2.

**Step IV-1:** For the input cell $I_i$ which is incident to $u_{ki}$ ( $| u_{ki} | = u^*$ ), if the input cell $I_i$ is included in the fuzzy cell group, go to Step IV-1-F; whereas in the crisp cell group, go to Step IV-1-C.

**Step IV-1-F:** If $SO_k \geq 0$, select one pattern of the fuzzy cell group which has the maximum value of $SO_k$ among (+1,-1,-1), (+1,+1,-1) and (+1,+1,+1). Conversely, If $SO_k < 0$, select one pattern which has the minimum value of $SO_k$. Go to Step V.

**Step IV-1-C:** If $SO_k \geq 0$, select one pattern of the crisp cell group which has the maximum value of $SO_k$ in (+1,+1,+1) and (-1,-1,-1). Conversely, If $SO_k < 0$, select one pattern which has the minimum value of $SO_k$. Go to Step V.

**Step IV-2:** Let $w^*$ be he maximum value of $| w_{ji} |$ which is an absolute value of the weight of the connections between the intermediate cell $H_j$ which is incident to $v_{kj}$ ( $| v_{kj} |$ = $v^*$ ) and the input cell $I_i$ whose activation value is 0. Select the input cell $I_i$ which is incident to the connection $w_{ji}$ ( $| w_{ji} | = w^*$ ). If the input cell $I_i$ is included in the fuzzy cell group, go to Step IV-2-F; whereas in the crisp cell group, go to Step IV-2-C.

**Step IV-2-F:** If $SH_j \geq 0$, select one pattern of the fuzzy cell group which has the maximum value of $SH_j$ among (+1,-1,-1), (+1,+1,-1) and (+1,+1,+1). Conversely, If $SH_j < 0$, select one pattern which has the minimum value of $SH_j$. Go to Step V.

**Step IV-2-C:** If $SH_j \geq 0$, select one pattern of the crisp cell group which has the maximum value of $SH_j$ in (+1,+1,+1) and (-1,-1,-1). Conversely, If $SH_j < 0$, select one pattern which has the minimum value of $SH_j$. Go to Step V.

**Step V (Extraction of framework of If-then Rules):** If the value of $O_k$ is determined, extract input items corresponding to a determined cell group as the propositions in an antecedent (If-part). Here, if the value of $O_k$ is +1, the consequence is set to "$O_k$ is *True*"; conversely if the value of $O_k$ is -1, the consequence is set to "$O_k$ is *False*". If multiple frameworks of If-Then rules with same antecedent and consequence are extracted, adopt one of them.

**Step VI (Termination condition of extraction algorithm for each output cell):** For output cell $O_k$, if there are any cell groups which are not selected yet; or for selected cell groups, there are any patterns which are not selected yet, go to Step II. Otherwise, go to Step VII.

**Step VII (Termination condition of whole extraction algorithm):** Repeat Steps II through VI stated above until the termination condition of extraction algorithm for each output cell is satisfied. If there are any output cell $O_k$ which are not selected yet in Step I, go to Step I. Otherwise, stop the whole extraction algorithm.

# 4  APPLICATION TO MEDICAL DIAGNOSIS

To prove the effectiveness and validity of the proposed neural expert system, we have developed neural expert systems for diagnosing hepatobiliary disorders (Yoshida et al., 1989 & 1990). We used a real medical database containing sex and the results of nine biochemical tests (e.g. GOT, GGT) of four hepatobiliary disorders, Alcoholic liver damage, Primary hepatoma, Liver cirrhosis and Cholelithiasis. The subjects consisted of 536 patients who were admitted to a university-affiliated hospital. The patients were clinically and pathologically diagnosed by physicians. The subjects were randomly assigned to 373 training data and 163 test (external) data. Degree of abnormality of each biochemical item is represented by a fuzzy cell group which consists of three input cells. There are four output cells. Each output cell corresponds to a hepatobiliary disorder. Fifty thousand iterations in learning process of Pocket Algorithm was performed for each

output cell. The diagnosis criteria is the same as that employed in (Yoshida et al. 1989). After learning by using training data from 345 patients, the fuzzy neural network correctly diagnosed 75.5% of test (external) data from 163 previously unseen patients and correctly diagnosed 100% of the training data. Conversely, the diagnostic accuracy of the linear discriminant analysis was 65.0% of the test data and 68.4% of the training data. The proposed fuzzy neural network showed significantly higher diagnostic accuracy in training data and also had substantially higher diagnostic accuracy in test data than those of linear discriminant analysis. We extracted 48 general fuzzy If-Then rules for diagnosing hepatobiliary disorders by using the proposed algorithm given in section 3.2. The number of rules for comfirming diseases are 12 and the those for excluding diseases are 36. Hayashi and Nakai (1989) have proposed three kinds of reasoning methods using fuzzy If-Then rules with linguistic relative importance. In the present paper, we use the reasoning method-I for the evaluation of extracted fuzzy If-Then rules. Total diagnostic accuracy of the twelve extracted rules (four confirming rules and eight excluding rules) is 87.7%. We conclude that the present neural network knowledge base approach will be a promising and useful technique for generating practical knowledge bases from various databases. It should be noted that enhancement of interpretation capability of real data, and embodiment of implicit and/or subjective knowledge will lead to significant reduction of man power for knowledge acquisition in expert system development.

## Acknowledgements

The author wishes to thank Dr. Stephen I. Gallant, Dr. Katsumi Yoshida and Mr. Atsushi Imura for their valuable comments and discussions.

## Footnotes

*A part of this work was performed when the author was with the University of Alabama at Birmingham, Department of Computer and Information Sciences as a Visiting Associate Professor.

## References

Gallant, S.I. 1988 Connectionist Expert Systems, CACM, 31(2), 152-169

Gallant, S.I. & Hayashi, Y. 1990 A Neural Network Expert System with Confidence Measurements, Proc. of the Third Int. Conf. on Infor. Proc. and Mgt. of Uncertainty in Knowledge-based Systems, pp.3-5, Paris, July 2-6; Springer Edited Volume (in press)

Gallant, S.I. 1990 Perceptron-Based Learning Algorithms, IEEE Transactions on Neural Networks, 1(2), 179-191

Hayashi, Y. & Nakai, M. 1989 Reasoning Methods Using a Fuzzy Production Rule with Linguistic Relative Importance in an Antecedent, The Transactions of The Institute of Electrical Engineers of Japan (T. IEE Japan), 109-C(9), 661-668

Hayashi, Y. & Nakai, M. 1990 Automated Extraction of Fuzzy IF-THEN Rules Using Neural Networks, T. IEE Japan, 110-C(3), 198-206

Hayashi, Y., Imura, A. & Yoshida, K. 1990 A Neural Expert System under Uncertain Environments and Its Evaluation, Proc. of the 11th Knowledge and Intelligence System Symposium, pp.13-18, Tokyo

Negoita, C.V. 1985 Expert Systems and Fuzzy Systems: Benjamin Cummings Pub.

Yoshida, K., Hayashi, Y. & Imura, A. 1989 A Connectionist Expert System for Diagnosing Hepatobiliary Disorders," in MEDINFO89 (Proc. of the Sixth Conf. on Medical Informatics), B. Barber et al. eds.: North-Holland, 116-120

Yoshida, K., Hayashi, Y., Imura, A. & Shimada, N. 1990 Fuzzy Neural Expert System for Diagnosing Hepatobiliary Disorders, Proc. of the Int. Conf. on Fuzzy Logic & Neural Networks (IIZUKA '90), pp.539-543, Iizuka, Japan, July 20-24